# A Reduction from Apprenticeship Learning to Classification

**Umar Syed**[*]
Department of Computer and Information Science
University of Pennsylvania
Philadelphia, PA 19104
usyed@cis.upenn.edu

**Robert E. Schapire**
Department of Computer Science
Princeton University
Princeton, NJ 08540
schapire@cs.princeton.edu

## Abstract

We provide new theoretical results for apprenticeship learning, a variant of reinforcement learning in which the true reward function is unknown, and the goal is to perform well relative to an observed expert. We study a common approach to learning from expert demonstrations: using a classification algorithm to learn to imitate the expert's behavior. Although this straightforward learning strategy is widely-used in practice, it has been subject to very little formal analysis. We prove that, if the learned classifier has error rate $\epsilon$, the difference between the value of the apprentice's policy and the expert's policy is $O(\sqrt{\epsilon})$. Further, we prove that this difference is only $O(\epsilon)$ when the expert's policy is close to optimal. This latter result has an important practical consequence: Not only does imitating a near-optimal expert result in a better policy, but far fewer demonstrations are required to successfully imitate such an expert. This suggests an opportunity for substantial savings whenever the expert is known to be good, but demonstrations are expensive or difficult to obtain.

## 1  Introduction

*Apprenticeship learning* is a variant of reinforcement learning, first introduced by Abbeel & Ng [1] (see also [2, 3, 4, 5, 6]), designed to address the difficulty of correctly specifying the reward function in many reinforcement learning problems. The basic idea underlying apprenticeship learning is that a learning agent, called the *apprentice*, is able to observe another agent, called the *expert*, behaving in a Markov Decision Process (MDP). The goal of the apprentice is to learn a policy that is at least as good as the expert's policy, relative to an *unknown* reward function. This is a weaker requirement than the usual goal in reinforcement learning, which is to find a policy that maximizes reward. The development of the apprenticeship learning framework was prompted by the observation that, although reward functions are often difficult to specify, demonstrations of good behavior by an expert are usually available. Therefore, by observing such a expert, one can infer information about the true reward function without needing to specify it.

Existing apprenticeship learning algorithms have a number of limitations. For one, they typically assume that the true reward function can be expressed as a linear combination of a set of known features. However, there may be cases where the apprentice is unwilling or unable to assume that the rewards have this structure. Additionally, most formulations of apprenticeship learning are actually *harder* than reinforcement learning; apprenticeship learning algorithms typically invoke reinforcement learning algorithms as subroutines, and their performance guarantees depend strongly on the quality of these subroutines. Consequently, these apprenticeship learning algorithms suffer from the same challenges of large state spaces, exploration vs. exploitation trade-offs, etc., as reinforcement

---

[*]Work done while the author was a student at Princeton University.

learning algorithms. This fact is somewhat contrary to the intuition that demonstrations from an expert — especially a good expert — should make the problem easier, not harder.

Another approach to using expert demonstrations that has received attention primarily in the empirical literature is to passively *imitate* the expert using a classification algorithm (see [7, Section 4] for a comprehensive survey). Classification is the most well-studied machine learning problem, and it is sensible to leverage our knowledge about this "easier" problem in order to solve a more "difficult" one. However, there has been little formal analysis of this straightforward learning strategy (the main recent example is Ross & Bagnell [8], discussed below). In this paper, we consider a setting in which an apprentice uses a classification algorithm to passively imitate an observed expert in an MDP, and we bound the difference between the value of the apprentice's policy and the value of the expert's policy in terms of the accuracy of the learned classifier. Put differently, we show that apprenticeship learning can be *reduced* to classification. The idea of reducing one learning problem to another was first proposed by Zadrozny & Langford [9].

Our main contributions in this paper are a pair of theoretical results. First, we show that the difference between the value of the apprentice's policy and the expert's policy is $O(\sqrt{\epsilon})$,[1] where $\epsilon \in (0, 1]$ is the error of the learned classifier. Secondly, and perhaps more interestingly, we extend our first result to prove that the difference in policy values is only $O(\epsilon)$ when the expert's policy is close to optimal. Of course, if one could perfectly imitate the expert, then naturally a near-optimal expert policy is preferred. But our result implies something further: that near-optimal experts are actually *easier* to imitate, in the sense that fewer demonstration are required to achieve the same performance guarantee. This has important practical consequences. If one is certain *a priori* that the expert is demonstrating good behavior, then our result implies that many fewer demonstrations need to be collected than if this were not the case. This can yield substantial savings when expert demonstrations are expensive or difficult to obtain.

## 2 Related Work

Several authors have reduced reinforcement learning to simpler problems. Bagnell *et al* [10] described an algorithm for constructing a good nonstationary policy from a sequence of good "one-step" policies. These policies are only concerned with maximizing reward collected in a single time step, and are learned with the help of observations from an expert. Langford & Zadrozny [11] reduced reinforcement learning to a sequence of classification problems (see also Blatt & Hero [12]), but these problems have an unusual structure, and the authors are only able to provide a small amount of guidance as to how data for these problems can be collected. Kakade & Langford [13] reduced reinforcement learning to regression, but required additional assumptions about how easily a learning algorithm can access the entire state space. Importantly, all this work makes the standard reinforcement learning assumptions that the true rewards are known, and that a learning algorithm is able to interact directly with the environment. In this paper we are interested in settings where the reward function is not known, and where the learning algorithm is limited to passively observing an expert. Concurrently to this work, Ross & Bagnell [8] have described an approach to reducing imitation learning to classification, and some of their analysis resembles ours. However, their framework requires somewhat more than passive observation of the expert, and is focused on improving the sensitivity of the reduction to the horizon length, not the classification error. They also assume that the expert follows a deterministic policy, and assumption we do not make.

## 3 Preliminaries

We consider a *finite-horizon* MDP, with horizon $H$. We will allow the state space $\mathcal{S}$ to be infinite, but assume that the action space $\mathcal{A}$ is finite. Let $\alpha$ be the initial state distribution, and $\theta$ the transition function, where $\theta(s, a, \cdot)$ specifies the next-state distribution from state $s \in \mathcal{S}$ under action $a \in \mathcal{A}$. The only assumption we make about the *unknown* reward function $R$ is that $0 \leq R(s) \leq R^{\max}$ for all states $s \in \mathcal{S}$, where $R^{\max}$ is a finite upper bound on the reward of any state.

We introduce some notation and definitions regarding policies. A policy $\pi$ is *stationary* if it is a mapping from states to distributions over actions. In this case, $\pi(s, a)$ denotes the probability of taking action $a$ in state $s$. Let $\Pi$ be the set of all stationary policies. A policy $\pi$ is *nonstationary* if it belongs to the set $\Pi^H = \Pi \times \cdots (H \text{ times}) \cdots \times \Pi$. In this case, $\pi_t(s, a)$ denotes the probability of taking action $a$ in state $s$ at time $t$. Also, if $\pi$ is nonstationary, then $\pi_t$ refers to the stationary policy that is equal to the $t^{th}$ component of $\pi$. A (stationary or nonstationary) policy $\pi$ is *deterministic* if each one of its action distributions is concentrated on a single action. If a deterministic policy $\pi$ is stationary, then $\pi(s)$ is the action taken in state $s$, and if $\pi$ is nonstationary, the $\pi_t(s)$ is the action taken in state $s$ at time $t$.

We define the value function $V_t^\pi(s)$ for a nonstationary policy $\pi$ at time $t$ as follows in the usual manner:

$$ V_t^\pi(s) \triangleq E \left[ \sum_{t'=t}^{H} R(s_{t'}) \ \Big| \ s_t = s, a_{t'} \sim \pi_{t'}(s_{t'}, \cdot), s_{t'+1} \sim \theta(s_{t'}, a_{t'}, \cdot) \right]. $$

So $V_t^\pi(s)$ is the expected cumulative reward for following policy $\pi$ when starting at state $s$ and time step $t$. Note that there are several value functions per nonstationary policy, one for each time step $t$. The value of a policy is defined to be $V(\pi) \triangleq E[V_1^\pi(s) \mid s \sim \alpha(\cdot)]$, and an optimal policy $\pi^*$ is one that satisfies $\pi^* \triangleq \arg\max_\pi V(\pi)$.

We write $\pi^E$ to denote the (possibly nonstationary) expert policy, and $V_t^E(s)$ as an abbreviation for $V_t^{\pi^E}(s)$. Our goal is to find a nonstationary apprentice policy $\pi^A$ such that $V(\pi^A) \geq V(\pi^E)$. Note that the values of these policies are with respect to the *unknown* reward function.

Let $D_t^\pi$ be the distribution on state-action pairs at time $t$ under policy $\pi$. In other words, a sample $(s, a)$ is drawn from $D_t^\pi$ by first drawing $s_1 \sim \alpha(\cdot)$, then following policy $\pi$ for time steps 1 through $t$, which generates a trajectory $(s_1, a_1, \ldots, s_t, a_t)$, and then letting $(s, a) = (s_t, a_t)$. We write $D_t^E$ as an abbreviation for $D_t^{\pi^E}$. In a minor abuse of notation, we write $s \sim D_t^\pi$ to mean: draw state-action pair $(s, a) \sim D_t^\pi$, and discard $a$.

## 4   Details and Justification of the Reduction

Our goal is to reduce apprenticeship learning to classification, so let us describe exactly how this reduction is defined, and also justify the utility of such a reduction.

In a classification problem, a learning algorithm is given a training set $\langle (x_1, y_1), \ldots, (x_m, y_m) \rangle$, where each labeled example $(x_i, y_i) \in \mathcal{X} \times \mathcal{Y}$ is drawn independently from a distribution $D$ on $\mathcal{X} \times \mathcal{Y}$. Here $\mathcal{X}$ is the example space and $\mathcal{Y}$ is the finite set of labels. The learning algorithm is also given the definition of a hypothesis class $\mathcal{H}$, which is a set of functions mapping $\mathcal{X}$ to $\mathcal{Y}$. The objective of the learning algorithm is to find a hypothesis $h \in \mathcal{H}$ such that the error $\Pr_{(x,y)\sim D}(h(x) \neq y)$ is small.

For our purposes, the hypothesis class $\mathcal{H}$ is said to be *PAC-learnable* if there exists a learning algorithm $A$ such that, whenever $A$ is given a training set of size $m = \text{poly}(\frac{1}{\delta}, \frac{1}{\epsilon})$, the algorithm runs for $\text{poly}(\frac{1}{\delta}, \frac{1}{\epsilon})$ steps and outputs a hypothesis $\hat{h} \in \mathcal{H}$ such that, with probability at least $1 - \delta$, we have $\Pr_{(x,y)\sim D}\left( \hat{h}(x) \neq y \right) \leq \epsilon^*_{\mathcal{H},D} + \epsilon$. Here $\epsilon^*_{\mathcal{H},D} = \inf_{h\in\mathcal{H}} \Pr_{(x,y)\sim D}(h(x) \neq y)$ is the error of the best hypothesis in $\mathcal{H}$. The expression $\text{poly}(\frac{1}{\delta}, \frac{1}{\epsilon})$ will typically also depend on other quantities, such as the number of labels $|\mathcal{Y}|$ and the *VC-dimension* of $\mathcal{H}$ [14], but this dependence is not germane to our discussion.

The existence of PAC-learnable hypothesis classes is the reason that reducing apprenticeship learning to classification is a sensible endeavor. Suppose that the apprentice observes $m$ independent trajectories from the expert's policy $\pi^E$, where the $i$th trajectory is a sequence $\left( s_1^i, a_1^i, \ldots, s_H^i, a_H^i \right)$. The key is to note that each $(s_t^i, a_t^i)$ can be viewed as an independent sample from the distribution $D_t^E$. Now consider a PAC-learnable hypothesis class $\mathcal{H}$, where $\mathcal{H}$ contains a set of functions mapping the state space $\mathcal{S}$ to the finite action space $\mathcal{A}$. If $m = \text{poly}(\frac{1}{H\delta}, \frac{1}{\epsilon})$, then for each time step $t$, the apprentice can use a PAC learning algorithm for $\mathcal{H}$ to learn a hypothesis $\hat{h}_t \in \mathcal{H}$ such that, with probability at least $1 - \frac{1}{H\delta}$, we have $\Pr_{(s,a)\sim D_t^E}\left( \hat{h}_t(s) \neq a \right) \leq \epsilon^*_{\mathcal{H}, D_t^E} + \epsilon$. And by the union

bound, this inequality holds for *all* $t$ with probability at least $1 - \delta$. If each $\epsilon^*_{\mathcal{H}, D^E_t} + \epsilon$ is small, then a natural choice for the apprentice's policy $\pi^A$ is to set $\pi^A_t = \hat{h}_t$ for all $t$. This policy uses the learned classifiers to imitate the behavior of the expert.

In light of the preceding discussion, throughout the remainder of this paper we make the following assumption about the apprentice's policy.

**Assumption 1.** *The apprentice policy $\pi^A$ is a deterministic policy that satisfies $\Pr_{(s,a) \sim D^E_t}(\pi^A_t(s) \neq a) \leq \epsilon$ for some $\epsilon > 0$ and all time steps $t$.*

As we have shown, an apprentice policy satisfying Assumption 1 with small $\epsilon$ can be found with high probability, provided that expert's policy is well-approximated by a PAC-learnable hypothesis class and that the apprentice is given enough trajectories from the expert. A reasonable intuition is that the value of the policy $\pi^A$ in Assumption 1 is nearly as high as the value of the policy $\pi^E$; the remainder of this paper is devoted to confirming this intuition.

## 5 Guarantee for Any Expert

If the error rate $\epsilon$ in Assumption 1 is small, then the apprentice's policy $\pi^A$ closely imitates the expert's policy $\pi^E$, and we might hope that this implies that $V(\pi^A)$ is not much less than $V(\pi^E)$. This is indeed the case, as the next theorem shows.

**Theorem 1.** *If Assumption 1 holds, then $V(\pi^A) \geq V(\pi^E) - 2\sqrt{\epsilon} H^2 R^{\max}$.*

In a typical classification problem, it is assumed that the training and test examples are drawn from the same distribution. The main challenge in proving Theorem 1 is that this assumption does *not* hold for the classification problems to which we have reduced the apprenticeship learning problem. This is because, although each state-action pair $(s^i_t, a^i_t)$ appearing in an expert trajectory is distributed according to $D^E_t$, a state-action pair $(s_t, a_t)$ visited by the apprentice's policy may not follow this distribution, since the behavior of the apprentice prior to time step $t$ may not exactly match the expert's behavior. So our strategy for proving Theorem 1 will be to show that these differences do not cause the value of the apprentice policy to degrade too much relative to the value of the expert's policy.

Before proceeding, we will show that Assumption 1 implies a condition that is, for our purposes, more convenient.

**Lemma 1.** *Let $\hat{\pi}$ be a deterministic nonstationary policy. If $\Pr_{(s,a) \sim D^E_t}(\hat{\pi}_t(s) \neq a) \leq \epsilon$, then for all $\epsilon_1 \in (0, 1]$ we have $\Pr_{s \sim D^E_t}\left(\pi^E_t(s, \hat{\pi}_t(s)) \geq 1 - \epsilon_1\right) \geq 1 - \frac{\epsilon}{\epsilon_1}$*

*Proof.* Fix any $\epsilon_1 \in (0, 1]$, and suppose for contradiction that $\Pr_{s \sim D^E_t}\left(\pi^E_t(s, \hat{\pi}_t(s)) \geq 1 - \epsilon_1\right) < 1 - \frac{\epsilon}{\epsilon_1}$. Say that a state $s$ is *good* if $\pi^E_t(s, \hat{\pi}_t(s)) \geq 1 - \epsilon_1$, and that $s$ is *bad* otherwise. Then

$$
\begin{aligned}
\Pr_{(s,a) \sim D^E_t}(\hat{\pi}_t(s) = a) &= \Pr_{s \sim D^E_t}(s \text{ is good}) \cdot \Pr_{(s,a) \sim D^E_t}(\hat{\pi}_t(s) = a \mid s \text{ is good}) \\
&\quad + \Pr_{s \sim D^E_t}(s \text{ is bad}) \cdot \Pr_{(s,a) \sim D^E_t}(\hat{\pi}_t(s) = a \mid s \text{ is bad}) \\
&\leq \Pr_{s \sim D^E_t}(s \text{ is good}) \cdot 1 + (1 - \Pr_{s \sim D^E_t}(s \text{ is good})) \cdot (1 - \epsilon_1) \\
&= 1 - \epsilon_1(1 - \Pr_{s \sim D^E_t}(s \text{ is good})) \\
&< 1 - \epsilon
\end{aligned}
$$

where the first inequality holds because $\Pr_{(s,a) \sim D^E_t}(\hat{\pi}_t(s) = a \mid s \text{ is bad}) \leq 1 - \epsilon_1$, and the second inequality holds because $\Pr_{s \sim D^E_t}(s \text{ is good}) < 1 - \frac{\epsilon}{\epsilon_1}$. This chain of inequalities clearly contradicts the assumption of the lemma. $\square$

The next two lemmas are the main tools used to prove Theorem 1. In the proofs of these lemmas, we write $\overline{sa}$ to denote a trajectory, where $\overline{sa} = (\bar{s}_1, \bar{a}_1, \ldots, \bar{s}_H, \bar{a}_H) \in (\mathcal{S} \times \mathcal{A})^H$. Also, let $dP_\pi$ denote the probability measure induced on trajectories by following policy $\pi$, and let $R(\overline{sa}) = \sum_{t=1}^{H} R(\bar{s}_t)$

denote the sum of the rewards of the states in trajectory $\overline{sa}$. Importantly, using these definitions we have

$$V(\pi) = \int_{\overline{sa}} R(\overline{sa})dP_\pi.$$

The next lemma proves that if a deterministic policy "almost" agrees with the expert's policy $\pi^E$ in every state and time step, then its value is not much worse the value of $\pi^E$.

**Lemma 2.** *Let $\hat{\pi}$ be a deterministic nonstationary policy. If for all states $s$ and time steps $t$ we have $\pi_t^E(s, \hat{\pi}_t(s)) \geq 1 - \epsilon$ then $V(\hat{\pi}) \geq V(\pi^E) - \epsilon H^2 R^{\max}$.*

*Proof.* Say a trajectory $\overline{sa}$ is *good* if it is "consistent" with $\hat{\pi}$ — that is, $\hat{\pi}(\bar{s}_t) = \bar{a}_t$ for all time steps $t$ — and that $\overline{sa}$ is *bad* otherwise. We have

$$
\begin{aligned}
V(\pi^E) &= \int_{\overline{sa}} R(\overline{sa})dP_{\pi^E} \\
&= \int_{\overline{sa} \text{ good}} R(\overline{sa})dP_{\pi^E} + \int_{\overline{sa} \text{ bad}} R(\overline{sa})dP_{\pi^E} \\
&\leq \int_{\overline{sa} \text{ good}} R(\overline{sa})dP_{\pi^E} + \epsilon H^2 R^{\max} \\
&\leq \int_{\overline{sa} \text{ good}} R(\overline{sa})dP_{\hat{\pi}} + \epsilon H^2 R^{\max} \\
&= V(\hat{\pi}) + \epsilon H^2 R^{\max}
\end{aligned}
$$

where the first inequality holds because, by the union bound, $P_{\pi^E}$ assigns at most an $\epsilon H$ fraction of its measure to bad trajectories, and the maximum reward of a trajectory is $HR^{\max}$. The second inequality holds because good trajectories are assigned at least as much measure by $P_{\hat{\pi}}$ as by $P_{\pi^E}$, because $\hat{\pi}$ is deterministic. $\qquad\square$

The next lemma proves a slightly different statement than Lemma 2: If a policy exactly agrees with the expert's policy $\pi^E$ in "almost" every state and time step, then its value is not much worse the value of $\pi^E$.

**Lemma 3.** *Let $\hat{\pi}$ be a nonstationary policy. If for all time steps $t$ we have $\Pr_{s \sim D_t^E}\left(\hat{\pi}_t(s, \cdot) = \pi_t^E(s, \cdot)\right) \geq 1 - \epsilon$ then $V(\hat{\pi}) \geq V(\pi^E) - \epsilon H^2 R^{\max}$.*

*Proof.* Say a trajectory $\overline{sa}$ is *good* if $\pi_t^E(\bar{s}_t, \cdot) = \hat{\pi}_t(\bar{s}_t, \cdot)$ for all time steps $t$, and that $\overline{sa}$ is *bad* otherwise. We have

$$
\begin{aligned}
V(\hat{\pi}) &= \int_{\overline{sa}} R(\overline{sa})dP_{\hat{\pi}} \\
&= \int_{\overline{sa} \text{ good}} R(\overline{sa})dP_{\hat{\pi}} + \int_{\overline{sa} \text{ bad}} R(\overline{sa})dP_{\hat{\pi}} \\
&= \int_{\overline{sa} \text{ good}} R(\overline{sa})dP_{\pi^E} + \int_{\overline{sa} \text{ bad}} R(\overline{sa})dP_{\hat{\pi}} \\
&= \int_{\overline{sa}} R(\overline{sa})dP_{\pi^E} - \int_{\overline{sa} \text{ bad}} R(\overline{sa})dP_{\pi^E} + \int_{\overline{sa} \text{ bad}} R(\overline{sa})dP_{\hat{\pi}} \\
&\geq V(\pi^E) - \epsilon H^2 R^{\max} + \int_{\overline{sa} \text{ bad}} R(\overline{sa})dP_{\hat{\pi}} \\
&\geq V(\pi^E) - \epsilon H^2 R^{\max}.
\end{aligned}
$$

The first inequality holds because, by the union bound, $P_{\pi^E}$ assigns at most an $\epsilon H$ fraction of its measure to bad trajectories, and the maximum reward of a trajectory is $HR^{\max}$. The second inequality holds by our assumption that all rewards are nonnegative. $\qquad\square$

We are now ready to combine the previous lemmas and prove Theorem 1.

*Proof of Theorem 1.* Since the apprentice's policy $\pi^A$ satisfies Assumption 1, by Lemma 1 we can choose any $\epsilon_1 \in (0, 1]$ and have

$$\mathrm{Pr}_{s \sim D_t^E} \left( \pi_t^E(s, \pi_t^A(s)) \geq 1 - \epsilon_1 \right) \geq 1 - \tfrac{\epsilon}{\epsilon_1}.$$

Now construct a "dummy" policy $\hat{\pi}$ as follows: For all time steps $t$, let $\hat{\pi}_t(s, \cdot) = \pi_t^E(s, \cdot)$ for any state $s$ where $\pi_t^E(s, \pi_t^A(s)) \geq 1 - \epsilon_1$. On all other states, let $\hat{\pi}_t(s, \pi_t^A(s)) = 1$. By Lemma 2

$$V(\pi^A) \geq V(\hat{\pi}) - \epsilon_1 H^2 R^{\mathrm{max}}$$

and by Lemma 3

$$V(\hat{\pi}) \geq V(\pi^E) - \frac{\epsilon}{\epsilon_1} H^2 R^{\mathrm{max}}.$$

Combining these inequalities yields

$$V(\pi^A) \geq V(\pi^E) - \left( \epsilon_1 + \frac{\epsilon}{\epsilon_1} \right) H^2 R^{\mathrm{max}}.$$

Since $\epsilon_1$ was chosen arbitrarily, we set $\epsilon_1 = \sqrt{\epsilon}$, which maximizes this lower bound. $\quad\square$

## 6 Guarantee for Good Expert

Theorem 1 makes no assumptions about the value of the expert's policy. However, in many cases it may be reasonable to assume that the expert is following a near-optimal policy (indeed, if she is not, then we should question the decision to select her as an expert). The next theorem shows that the dependence of $V(\pi^A)$ on the classification error $\epsilon$ is significantly better when the expert is following a near-optimal policy.

**Theorem 2.** *If Assumption 1 holds, then* $V(\pi^A) \geq V(\pi^E) - \left( 4\epsilon H^3 R^{\mathrm{max}} + \Delta_{\pi^E} \right)$, *where* $\Delta_{\pi^E} \triangleq V(\pi^*) - V(\pi^E)$ *is the* suboptimality *of the expert's policy* $\pi^E$.

Note that the bound in Theorem 2 varies with $\epsilon$ and not with $\sqrt{\epsilon}$. We can interpret this bound as follows: If our goal is to learn an apprentice policy whose value is within $\Delta_{\pi^E}$ of the expert policy's value, we can double our progress towards that goal by halving the classification error rate. On the other hand, Theorem 2 suggests that the error rate must be reduced by a factor of four.

To see why a near-optimal expert policy should yield a weaker dependence on $\epsilon$, consider an expert policy $\pi^E$ that is an optimal policy, but in every state $s \in \mathcal{S}$ selects one of two actions $a_1^s$ and $a_2^s$ uniformly at random. A deterministic apprentice policy $\pi^A$ that closely imitates the expert will either set $\pi^A(s) = a_1^s$ or $\pi^A(s) = a_2^s$, but in either case the classification error will not be less than $\frac{1}{2}$. However, since $\pi^E$ is optimal, both actions $a_1^s$ and $a_2^s$ must be optimal actions for state $s$, and so the apprentice policy $\pi^A$ will be optimal as well.

Our strategy for proving Theorem 2 is to replace Lemma 2 with a different result — namely, Lemma 6 below — that has a much weaker dependence on the classification error $\epsilon$ when $\Delta_{\pi^E}$ is small.

To help us prove Lemma 6, we will first need to define several useful policies. The next several definitions will be with respect to an arbitrary nonstationary *base policy* $\pi^B$; in the proof of Theorem 2, we will make a particular choice for the base policy.

Fix a deterministic nonstationary policy $\pi^{B,\epsilon}$ that satisfies

$$\pi_t^B(s, \pi_t^{B,\epsilon}(s)) \geq 1 - \epsilon$$

for some $\epsilon \in (0, 1]$ and all states $s$ and time steps $t$. Such a policy always exists by letting $\epsilon = 1$, but if $\epsilon$ is close to zero, then $\pi^{B,\epsilon}$ is a deterministic policy that "almost" agrees with $\pi^B$ in every state and time step. Of course, depending on the choice of $\pi^B$, a policy $\pi^{B,\epsilon}$ may not exist for small $\epsilon$, but let us set aside that concern for the moment; in the proof of Theorem 2, the base policy $\pi^B$ will be chosen so that $\epsilon$ can be as small as we like.

Having thus defined $\pi^{B,\epsilon}$, we define $\pi^{B \setminus \epsilon}$ as follows: For all states $s \in \mathcal{S}$ and time steps $t$, if $\pi_t^B(s, \pi^{B,\epsilon}(s)) < 1$, then let

$$\pi_t^{B \setminus \epsilon}(s, a) = \begin{cases} 0 & \text{if } \pi_t^{B,\epsilon}(s) = a \\[2ex] \dfrac{\pi_t^B(s, a)}{\sum_{a' \neq \pi_t^{B,\epsilon}(s)} \pi_t^B(s, a')} & \text{otherwise} \end{cases}$$

for all actions $a \in \mathcal{A}$, and otherwise let $\pi_t^{B \backslash \epsilon}(s, a) = \frac{1}{|\mathcal{A}|}$ for all $a \in \mathcal{A}$. In other words, in each state $s$ and time step $t$, the distribution $\pi_t^{B \backslash \epsilon}(s, \cdot)$ is obtained by proportionally redistributing the probability assigned to action $\pi_t^{B, \epsilon}(s)$ by the distribution $\pi_t^B(s, \cdot)$ to all other actions. The case where $\pi_t^B(s, \cdot)$ assigns all probability to action $\pi_t^{B, \epsilon}(s)$ is treated specially, but as will be clear from the proof of Lemma 4, it is actually immaterial how the distribution $\pi_t^{B \backslash \epsilon}(s, \cdot)$ is defined in these cases; we choose the uniform distribution for definiteness.

Let $\pi^{B+}$ be a deterministic policy defined by

$$\pi_t^{B+}(s) = \arg\max_a E\left[V_{t+1}^{\pi^B}(s') \,\middle|\, s' \sim \theta(s, a, \cdot)\right]$$

for all states $s \in \mathcal{S}$ and time steps $t$. In other words, $\pi_t^{B+}(s)$ is the best action in state $s$ at time $t$, assuming that the policy $\pi^B$ is followed thereafter.

The next definition requires the use of *mixed policies*. A mixed policy consists of a finite set of deterministic nonstationary policies, along with a distribution over those policies; the mixed policy is followed by drawing a single policy according to the distribution in the initial time step, and following that policy exclusively thereafter. More formally, a mixed policy is defined by a set of ordered pairs $\{(\pi^i, \lambda(i))\}_{i=1}^N$ for some finite $N$, where each *component policy* $\pi^i$ is a deterministic nonstationary policy, $\sum_{i=1}^N \lambda(i) = 1$ and $\lambda(i) \geq 0$ for all $i \in [N]$.

We define a mixed policy $\tilde{\pi}^{B, \epsilon, +}$ as follows: For each component policy $\pi^i$ and each time step $t$, either $\pi_t^i = \pi_t^{B, \epsilon}$ or $\pi_t^i = \pi_t^{B+}$. There is one component policy for each possible choice; this yields $N = 2^{|H|}$ component policies. And the probability $\lambda(i)$ assigned to each component policy $\pi^i$ is $\lambda(i) = (1 - \epsilon)^{k(i)} \epsilon^{H - k(i)}$, where $k(i)$ is the number of times steps $t$ for which $\pi_t^i = \pi_t^{B, \epsilon}$.

Having established these definitions, we are now ready to prove several lemmas that will help us prove Theorem 2.

**Lemma 4.** $V(\tilde{\pi}^{B, \epsilon, +}) \geq V(\pi^B)$.

*Proof.* The proof will be by backwards induction on $t$. Clearly $V_H^{\tilde{\pi}^{B, \epsilon, +}}(s) = V_H^{\pi^B}(s)$ for all states $s$, since the value function $V_H^\pi$ for any policy $\pi$ depends only on the reward function $R$. Now suppose for induction that $V_{t+1}^{\tilde{\pi}^{B, \epsilon, +}}(s) \geq V_{t+1}^{\pi^B}(s)$ for all states $s$. Then for all states $s$

$$
\begin{aligned}
V_t^{\tilde{\pi}^{B, \epsilon, +}}(s) &= R(s) + E\left[V_{t+1}^{\tilde{\pi}^{B, \epsilon, +}}(s') \,\middle|\, a' \sim \tilde{\pi}_t^{B, \epsilon, +}(s, \cdot), s' \sim \theta(s, a', \cdot)\right] \\
&\geq R(s) + E\left[V_{t+1}^{\pi^B}(s') \,\middle|\, a' \sim \tilde{\pi}_t^{B, \epsilon, +}(s, \cdot), s' \sim \theta(s, a', \cdot)\right] \\
&= R(s) + (1 - \epsilon) E\left[V_{t+1}^{\pi^B}(s') \,\middle|\, s' \sim \theta(s, \pi_t^{B, \epsilon}(s), \cdot)\right] + \epsilon E\left[V_{t+1}^{\pi^B}(s') \,\middle|\, s' \sim \theta(s, \pi_t^{B+}(s), \cdot)\right] \\
&\geq R(s) + \pi_t^B(s, \pi_t^{B, \epsilon}(s)) \cdot E\left[V_{t+1}^{\pi^B}(s') \,\middle|\, s' \sim \theta(s, \pi_t^{B, \epsilon}(s), \cdot)\right] \\
&\quad + \left(1 - \pi_t^B(s, \pi_t^{B, \epsilon}(s))\right) \cdot E\left[V_{t+1}^{\pi^B}(s') \,\middle|\, s' \sim \theta(s, \pi_t^{B+}(s), \cdot)\right] \\
&\geq R(s) + \pi_t^B(s, \pi_t^{B, \epsilon}(s)) \cdot E\left[V_{t+1}^{\pi^B}(s') \,\middle|\, s' \sim \theta(s, \pi_t^{B, \epsilon}(s), \cdot)\right] \\
&\quad + \left(1 - \pi_t^B(s, \pi_t^{B, \epsilon}(s))\right) \cdot E\left[V_{t+1}^{\pi^B}(s') \,\middle|\, a' \sim \pi_t^{B \backslash \epsilon}(s, \cdot), s' \sim \theta(s, a', \cdot)\right] \\
&= R(s) + E\left[V_{t+1}^{\pi^B}(s') \,\middle|\, a' \sim \pi_t^B(s), s' \sim \theta(s, a', \cdot)\right] \\
&= V_t^{\pi^B}(s).
\end{aligned}
$$

The first equality holds for all policies $\pi$, and follows straightforwardly from the definition of $V_t^\pi$. The rest of the derivation uses, in order: the inductive hypothesis; the definition of $\tilde{\pi}^{B, \epsilon, +}$; property of $\pi^{B, \epsilon}$ and the fact that $\pi_t^{B+}(s)$ is the best action with respect to $V_{t+1}^{\pi^B}$; the fact that $\pi_t^{B+}(s)$ is the best action with respect to $V_{t+1}^{\pi^B}$; the definition of $\pi^{B \backslash \epsilon}$; the definition of $V_t^{\pi^B}(s)$. $\square$

**Lemma 5.** $V(\tilde{\pi}^{B, \epsilon, +}) \leq (1 - \epsilon H) V(\pi^{B, \epsilon}) + \epsilon H V(\pi^*)$.

*Proof.* Since $\tilde{\pi}^{B,\epsilon,+}$ is a mixed policy, by the linearity of expectation we have

$$V(\tilde{\pi}^{B,\epsilon,+}) = \sum_{i=1}^{N} \lambda(i) V(\pi^i)$$

where each $\pi^i$ is a component policy of $\tilde{\pi}^{B,\epsilon,+}$ and $\lambda(i)$ is its associated probability. Therefore

$$
\begin{aligned}
V(\tilde{\pi}^{B,\epsilon,+}) &= \sum_i \lambda(i) V(\pi^i) \\
&\leq (1-\epsilon)^H V(\pi^{B,\epsilon}) + (1-(1-\epsilon)^H) V(\pi^*) \\
&\leq (1-\epsilon H) V(\pi^{B,\epsilon}) + \epsilon H V(\pi^*).
\end{aligned}
$$

Here we used the fact that probability $(1-\epsilon)^H \geq 1 - \epsilon H$ is assigned to a component policy that is identical to $\pi^{B,\epsilon}$, and the value of any component policy is at most $V(\pi^*)$. $\qquad\square$

**Lemma 6.** *If $\epsilon < \frac{1}{H}$, then $V(\pi^{B,\epsilon}) \geq V(\pi^B) - \frac{\epsilon H}{1-\epsilon H}\Delta_{\pi^B}$.*

*Proof.* Combining Lemmas 4 and 5 yields

$$(1-\epsilon H)V(\pi^{B,\epsilon}) + \epsilon H V(\pi^*) \geq V(\pi^B).$$

And via algebraic manipulation we have

$$
\begin{aligned}
& (1-\epsilon H)V(\pi^{B,\epsilon}) + \epsilon H V(\pi^*) \geq V(\pi^B) \\
\Rightarrow\ & (1-\epsilon H)V(\pi^{B,\epsilon}) \geq (1-\epsilon H)V(\pi^B) + \epsilon H V(\pi^B) - \epsilon H V(\pi^*) \\
\Rightarrow\ & (1-\epsilon H)V(\pi^{B,\epsilon}) \geq (1-\epsilon H)V(\pi^B) - \epsilon H \Delta_{\pi^B} \\
\Rightarrow\ & V(\pi^{B,\epsilon}) \geq V(\pi^B) - \frac{\epsilon H}{1-\epsilon H}\Delta_{\pi^B}.
\end{aligned}
$$

In the last line, we were able to divide by $(1-\epsilon H)$ without changing the direction of the inequality because of our assumption that $\epsilon < \frac{1}{H}$. $\qquad\square$

We are now ready to combine the previous lemmas and prove Theorem 2.

*Proof of Theorem 2.* Since the apprentice's policy $\pi^A$ satisfies Assumption 1, by Lemma 1 we can choose any $\epsilon_1 \in (0, \frac{1}{H})$ and have

$$\Pr_{s \sim D_t^E}\left(\pi_t^E(s, \pi_t^A(s)) \geq 1 - \epsilon_1\right) \geq 1 - \frac{\epsilon}{\epsilon_1}.$$

As in the proof of Theorem 1, let us construct a "dummy" policy $\hat{\pi}$ as follows: For all time steps $t$, let $\hat{\pi}_t(s,\cdot) = \pi_t^E(s,\cdot)$ for any state $s$ where $\pi_t^E(s, \pi_t^A(s)) \geq 1 - \epsilon_1$. On all other states, let $\hat{\pi}_t(s, \pi_t^A(s)) = 1$. By Lemma 3 we have

$$V(\hat{\pi}) \geq V(\pi^E) - \frac{\epsilon}{\epsilon_1} H^2 R^{\max}. \tag{1}$$

Substituting $V(\pi^E) = V(\pi^*) - \Delta_{\pi^E}$ and $V(\hat{\pi}) = V(\pi^*) - \Delta_{\hat{\pi}}$ and rearranging yields

$$\Delta_{\hat{\pi}} \leq \Delta_{\pi^E} + \frac{\epsilon}{\epsilon_1} H^2 R^{\max}. \tag{2}$$

Now observe that, if we set the base policy $\pi^B = \hat{\pi}$, then by definition $\pi^A$ is a valid choice for $\pi^{B,\epsilon_1}$. And since $\epsilon_1 < \frac{1}{H}$ we have

$$
\begin{aligned}
V(\pi^A) &\geq V(\hat{\pi}) - \frac{\epsilon_1 H}{1 - \epsilon_1 H}\Delta_{\hat{\pi}} \\
&\geq V(\hat{\pi}) - \frac{\epsilon_1 H}{1 - \epsilon_1 H}\left(\Delta_{\pi^E} + \frac{\epsilon}{\epsilon_1} H^2 R^{\max}\right) \\
&\geq V(\pi^E) - \frac{\epsilon}{\epsilon_1} H^2 R^{\max} - \frac{\epsilon_1 H}{1 - \epsilon_1 H}\left(\Delta_{\pi^E} + \frac{\epsilon}{\epsilon_1} H^2 R^{\max}\right)
\end{aligned} \tag{3}
$$

where we used Lemma 6, (2) and (1), in that order. Letting $\epsilon_1 = \frac{1}{2H}$ proves the theorem. $\qquad\square$

## Footnotes

[1]The big-O notation is concealing a polynomial dependence on other problem parameters. We give exact bounds in the body of the paper.

## References

[1] Pieter Abbeel and Andrew Ng. Apprenticeship learning via inverse reinforcement learning. In *Proceedings of the 21st International Conference on Machine Learning*, 2004.

[2] P Abbeel and A Y Ng. Exploration and apprenticeship learning in reinforcement learning. In *Proceedings of the 22nd International Conference on Machine Learning*, 2005.

[3] Nathan D. Ratliff, J. Andrew Bagnell, and Martin A. Zinkevich. Maximum margin planning. In *Proceedings of the 23rd International Conference on Machine Learning*, 2006.

[4] Umar Syed and Robert E. Schapire. A game-theoretic approach to apprenticeship learning. In *Advances in Neural Information Processing Systems 20*, 2008.

[5] J. Zico Kolter, Pieter Abbeel, and Andrew Ng. Hierarchical apprenticeship learning with application to quadruped locomotion. In *Advances in Neural Information Processing Systems 20*, 2008.

[6] Umar Syed and Robert E. Schapire. Apprenticeship learning using linear programming. In *Proceedings of the 25th International Conference on Machine Learning*, 2008.

[7] Brenna D. Argall, Sonia Chernova, Manuela Veloso, and Brett Browning. A survey of robot learning from demonstration. *Robotics and Autonomous Systems*, 57(5):469–483, 2009.

[8] Stéphane Ross and J. Andrew Bagnell. Efficient reduction for imitation learning. In *AISTATS*, 2010.

[9] Bianca Zadrozny, John Langford, and Naoki Abe. Cost-sensitive learning by cost-proportionate example weighting. In *Proceedings of the Third IEEE International Conference on Data Mining*, 2003.

[10] J. Andrew Bagnell, Sham Kakade, Andrew Y. Ng, and Jeff Schneider. Policy search by dynamic programming. In *Advances in Neural Information Processing Systems 15*, 2003.

[11] John Langford and Bianca Zadrozny. Relating reinforcement learning performance to classification performance. In *Proceedings of the 22nd International Conference on Machine Learning*, 2005.

[12] Doron Blatt and Alfred Hero. From weighted classification to policy search. In *Advances in Neural Information Processing Systems 18*, pages 139–146, 2006.

[13] Sham Kakade and John Langford. Approximately optimal approximate reinforcement learning. In *Proceedings 19th International Conference on Machine Learning*, 2002.

[14] V. N. Vapnik and A. Chervonenkis. On the uniform convergence of relative frequencies of events to their probabilities. *Theory of Probability and Its Applications*, 16:264–280, 1971.

